# Exploiting weakly-labeled Web images to improve object classification: a domain adaptation approach

**Alessandro Bergamo**        **Lorenzo Torresani**
Computer Science Department
Dartmouth College
Hanover, NH 03755, U.S.A.
{aleb, lorenzo}@cs.dartmouth.edu

## Abstract

Most current image categorization methods require large collections of manually annotated training examples to learn accurate visual recognition models. The time-consuming human labeling effort effectively limits these approaches to recognition problems involving a small number of different object classes. In order to address this shortcoming, in recent years several authors have proposed to learn object classifiers from weakly-labeled Internet images, such as photos retrieved by keyword-based image search engines. While this strategy eliminates the need for human supervision, the recognition accuracies of these methods are considerably lower than those obtained with fully-supervised approaches, because of the noisy nature of the labels associated to Web data.

In this paper we investigate and compare methods that learn image classifiers by combining very few manually annotated examples (e.g., 1-10 images per class) and a large number of weakly-labeled Web photos retrieved using keyword-based image search. We cast this as a domain adaptation problem: given a few strongly-labeled examples in a target domain (the manually annotated examples) and many source domain examples (the weakly-labeled Web photos), learn classifiers yielding small generalization error on the target domain. Our experiments demonstrate that, for the same number of strongly-labeled examples, our domain adaptation approach produces significant recognition rate improvements over the best published results (e.g., 65% better when using 5 labeled training examples per class) and that our classifiers are one order of magnitude faster to learn and to evaluate than the best competing method, despite our use of large weakly-labeled data sets.

## 1 Introduction

The last few years have seen a proliferation of human efforts to collect labeled image data sets for the purpose of training and evaluating visual recognition systems. Label information in these collections comes in different forms, ranging from simple object category labels to detailed semantic pixel-level segmentations. Examples include Caltech256 [14], and the Pascal VOC2010 data set [7]. In order to increase the variety and the number of labeled object classes, a few authors have designed online games and appealing software tools encouraging common users to participate in these image annotation efforts [23, 30]. Despite the tremendous research contribution brought by such attempts, even the largest labeled image collections today [6] are limited to a number of classes that is at least one order of magnitude smaller than the number of object categories that humans can recognize [3]. In order to overcome this limitation and in an attempt to build classifiers for arbitrary object classes, several authors have proposed systems that learn from weakly-labeled Internet photos [10, 9, 29, 20]. Most of these approaches rely on keyword-based image search engines to retrieve image examples of specified object classes. Unfortunately, while image search engines provide training examples

without the need of any human intervention, it is sufficient to type a few example keywords in Google or Bing image search to verify that often the majority of the retrieved images are only loosely related with the query concept. Most prior work has attempted to address this problem by means of outlier rejection mechanisms discarding irrelevant images from the retrieved results. However, despite the dynamic research activity in this area, weakly-supervised approaches today still yield significantly lower recognition accuracy than fully supervised object classifiers trained on clean data (see, e.g., results reported in [9, 29]).

In this paper we argue that the poor performance of models learned from weakly-labeled Internet data is not only due to undetected outliers contaminating the training data, but it is also a consequence of the statistical differences often present between Web images and the test data. Figure 1 shows sample images for some of the Caltech256 object categories versus the top six images retrieved by Bing using the class names as keywords[1]. Although a couple of outliers are indeed present in the Bing sets, the striking difference between the two collections is that even the relevant results in the Bing groups appear to be visually less homogeneous. For example, in the case of the classes shown in figure 1(a,b), while the Caltech256 groups contain only real photographs, the Bing counterparts include several cartoon drawings. In figure 1(c,d), each Caltech256 image contains only the object of interest while the pictures retrieved by Bing include extraneous items, such as people or faces, which act as distractors in the learning (this is particularly true when evaluating the classifiers on Caltech256, given that "faces" and "people" are separate categories in the data set). Furthermore, even when "irrelevant" results do occur in the retrieved images, they are rarely outliers detectable via simple coherence tests as there is often some consistency even among such photos. For example, polysemy — the capacity of one word to have multiple meanings — causes multiple visual clusters (as opposed to individual outliers) to appear in the Bing sets of figure 1(e,f) (the two clusters in (e) are due to the fact that the word "hawksbill" denotes both a crag in Arkansas as well as a type of sea turtle, while in the case of (f) the keyword "tricycle" retrieves images of both bicycles as well as motorcycles with three wheels; note, again, that Caltech256 contains for both classes only images corresponding to one of the words meanings and that "motorcycle" appears as a separate additional category). Finally, in some situations, different shooting distances or angles may produce completely unrelated views of the same object or scene: for example, the Bing set in 1(g) includes both aerial and ground views of Mars, which have very little in common visually.

Note that for most of the classes in figure 1 it is not clear a priori which are the "relevant" Internet images to be used for training until we compare them to the photos in the corresponding Caltech256 categories. In this paper we show that a few strongly-labeled examples from the test domain (e.g. a few Caltech256 images for the class of interest) are indeed sufficient to disambiguate this relevancy problem and to model the distribution differences between the weakly-labeled Internet data and the test application data, so as to significantly improve recognition performance on the test set.

The situation where the test data is drawn from a distribution that is related, but not identical, to the distribution of the training data has been widely studied in the field of machine learning and it is traditionally addressed using so-called "domain adaptation" methods. These techniques exploit ample availability of training data from a *source domain* to learn a model that works effectively in a related *target domain* for which only few training examples are available. More formally, let $p^t(X, Y)$ and $p^s(X, Y)$ be the distributions generating the target and the source data, respectively. Here, $X$ denotes the input (a random feature vector) and $Y$ the class (a discrete random variable). The domain adaptation problem arises whenever $p^t(X, Y)$ differs from $p^s(X, Y)$. In covariance shift, it is assumed that only the distributions of the input features differ in the two domain, i.e., $p^t(Y|X) = p^s(Y|X)$ but $p^t(X) \neq p^s(X)$. Note that, without adaptation, this may lead to poor classification in the target domain since a model learned from a large source training set will be trained to perform well in the dense source regions of $X$ which, under the covariance shift assumption, will generally be different from the dense regions of the target domain. Typically, covariance shift algorithms (e.g., [16]) address this problem by modeling the ratio $p^t(X)/p^s(X)$. Unfortunately, the much more common and challenging case is when the conditional distributions are different, i.e., $p^t(Y|X) \neq p^s(Y|X)$. When such differences are relatively small, however, knowledge gained by analyzing data in the source domain may still yield valuable information to perform prediction for test target data. This is precisely the scenario considered in this paper.

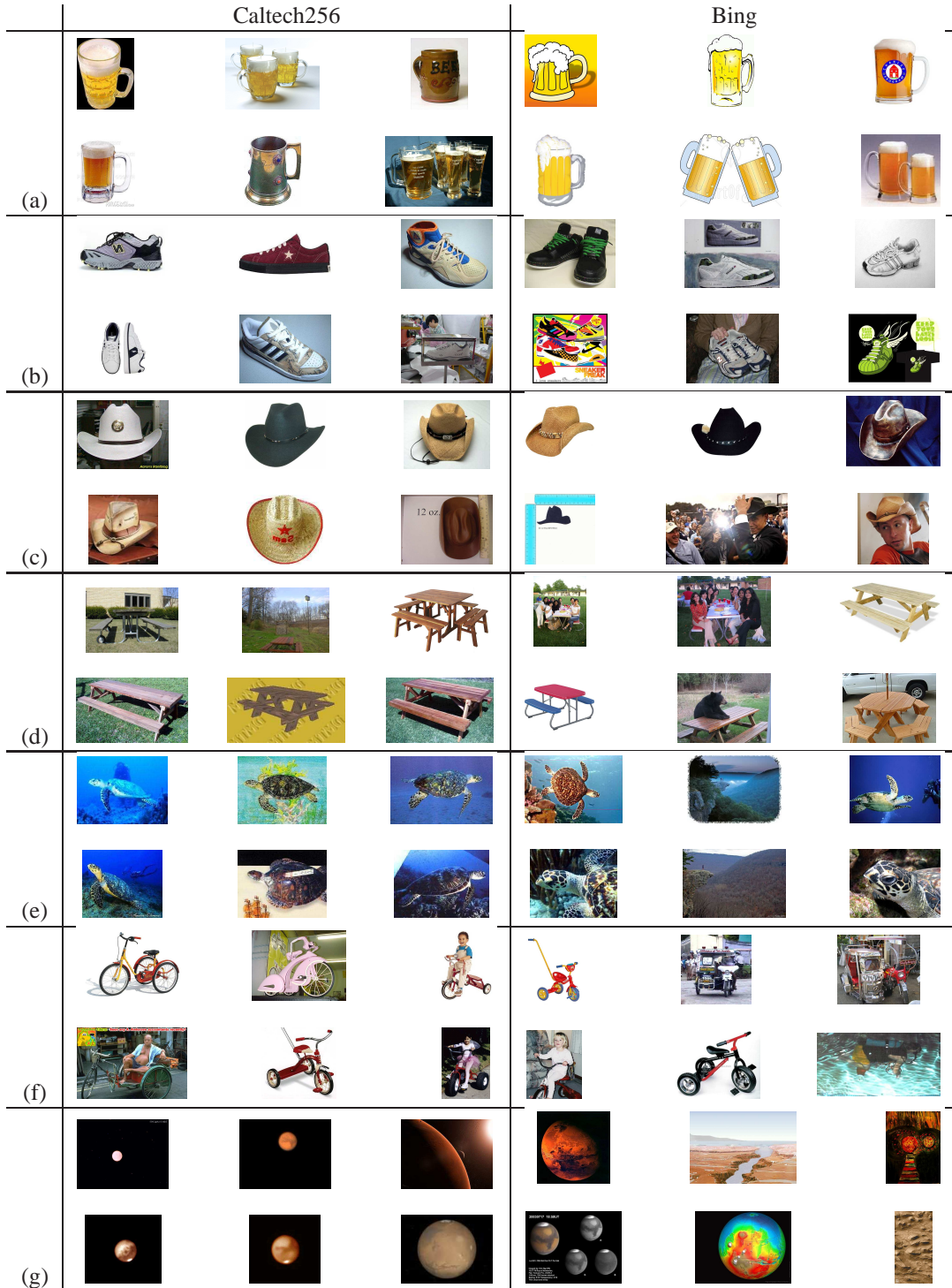

Figure 1: Images in Caltech256 for several categories and top results retrieved by Bing image search for the corresponding keywords. The Bing sets are both semantically and visually less coherent: presence of multiple objects in the same image, polysemy, caricaturization, as well as variations in viewpoints are some of the visual effects present in Internet images which cause significant data distribution differences between the Bing sets and the corresponding Caltech256 groups.

## 2 Relationship to other methods

Most of the prior work on learning visual models from image search has focused on the task of "cleaning up" Internet photos. For example, in the pioneering work of Fergus et al. [10], visual filters learned from image search were used to rerank photos on the basis of visual consistency. Subsequent approaches [2, 25, 20] have employed similar outlier rejection schemes to automatically construct clean(er) data sets of images for training and testing object classifiers. Even techniques aimed at learning explicit object classifiers from image search [9, 29] have identified outlier removal as the key-ingredient to improve recognition. In our paper we focus on another fundamental, yet largely ignored, aspect of the problem: we argue that the current poor performance of classification models learned from the Web is due to the distribution differences between Internet photos and image test examples. To the best of our knowledge we propose the first systematic empirical analysis of domain adaptation methods to address sample distribution differences in object categorization due to the use of weakly-labeled Web images as training data. We note that in work concurrent to our own, Saenko et al. [24] have also analyzed cross-domain adaptation of object classifiers. However, their work focuses on the statistical differences caused by varying lighting conditions (uncontrolled versus studio setups) and by images taken with different camera types (a digital SLR versus a webcam).

Transfer learning, also known as multi-task learning, is related to domain adaptation. In computer vision, transfer learning has been applied to a wide range of problems including object categorization (see, e.g., [21, 8, 22]). However, transfer learning addresses a different problem. In transfer learning there is a single distribution of the inputs $p(X)$ but there are multiple output variables $Y_1, \ldots, Y_T$, associated to $T$ distinct tasks (e.g., learning classifiers for different object classes). Typically, it is assumed that some relations exist among the tasks; for example, some common structure when learning classifiers $p(Y_1|X, \theta_1), \ldots, p(Y_T|X, \theta_T)$ can be enforced by assuming that the parameters $\theta_1, \ldots, \theta_T$ are generated from a shared prior $p(\theta)$. The fundamental difference is that in domain adaptation we have a single task but different domains, i.e., different sources of data.

As our approach relies on a mix of labeled and weakly-labeled images, it is loosely related to semi-supervised methods for object classification [15, 19]. Within this genre, the algorithm described in [11] is perhaps the closest to our work as it also relies on weakly-labeled Internet images. However, unlike our approach, these semi-supervised methods are designed to work in cases where the test examples and the training data are generated from the same distribution.

## 3 Approach overview

### 3.1 Experimental setup

Our objective is to evaluate domain adaptations methods on the task of object classification, using photos from a human-labeled data set as target domain examples and images retrieved by a keyword-based image search engine as examples of the source domain.

We used Caltech256 as the data set for the target domain since it is an established benchmark for object categorization and it contains a large number of classes (256) thus allowing us to average out performance variations due to especially easy or difficult categories. From each class, we randomly sampled $n_T$ images as target training examples, and other $m_T$ images as target test examples.

We formed the weakly-labeled source data by collecting the top $n_S$ images retrieved by Bing image search for each of the Caltech256 category text labels. Although it may have been possible to improve the relevancy of the image results for some of the classes by manually selecting less ambiguous search keywords, we chose to issue queries on the unchanged Caltech256 text class labels to avoid subjective alteration of the results. However, in order to ensure valid testing, we removed near duplicates of Caltech256 images from the source training set by a human-supervised process.

### 3.2 Feature representation and classification model

In order to study the effect of large weakly-labeled training sets on object recognition performance, we need a baseline system that achieves good performance on object categorization and that supports efficient learning and test evaluation. The current best published results on Caltech256 were obtained by a kernel combination classifier using 39 different feature kernels, one for each feature type [13]. However, since both training as well testing are computationally very expensive with this classifier, this model is unsuitable for our needs.

Instead, in this work we use as image representation the classeme features recently proposed by Torresani et al. [28]. This descriptor is particularly suitable for our task as it has been shown to yield near state-of-the-art results with simple linear support vector machines, which can be learned very efficiently even for large training sets. The descriptor measures the closeness of an image to a basis set of classes and can be used as an intermediate representation to learn classifiers for new classes. The basis classifiers of the classeme descriptor are learned from weakly-labeled data collected for a large and semantically broad set of attributes (the final descriptor contains 2659 attributes). To eliminate the risk of the test classes being already explicitly represented in the feature vector, in this work we removed from the descriptor 34 attributes, corresponding to categories related to Caltech256 classes. We use a binarized version of this descriptor obtained by thresholding to 0 the output of the attribute classifiers: this yields for each image a 2625-dimensional binary vector describing the predicted presence/absence of visual attributes in the photo. This binarization has been shown to yield very little degradation in recognition performance (see [28] for further details). We denote with $\boldsymbol{f}(\boldsymbol{x}) \in \{0,1\}^F$ the binary attribute vector extracted from image $\boldsymbol{x}$ with $F = 2625$.

Object class recognition is traditionally formulated as a multiclass classification problem: given a test image $\boldsymbol{x}$, predict the class label $y \in \{1, \ldots, K\}$ of the object present in it, where $K$ is the number of possible classes (in the case of Caltech256, $K = 256$). In this paper we implement multi-class classification using $K$ binary classifiers trained using the *one-versus-the-rest* scheme and perform prediction according to the *winner-take-all* strategy. The $k$-th binary classifier (distinguishing between class $k$ and the other classes) is trained on a target training set $\mathcal{D}_k^t$ and a collection $\mathcal{D}_k^s$ of weakly-labeled source training examples. $\mathcal{D}_k^t$ is formed by aggregating the Caltech256 training images of all classes, using the data from the $k$-th class as positive examples and the data from the remaining classes as negative examples, i.e. $\mathcal{D}_k^t = \{(\boldsymbol{f}_i^t, y_{i,k}^t)\}_{i=1}^{N_t}$ where $\boldsymbol{f}_i^t = \boldsymbol{f}(\boldsymbol{x}_i^t)$ denotes the feature vector of the $i$-th image, $N_t = (K \cdot n_t)$ is the total number of images in the strongly-labeled data set, and $y_{i,k}^t \in \{-1,1\}$ is 1 iff example $i$ belongs to class $k$. The source training set $\mathcal{D}_k^s = \{\boldsymbol{f}_{i,k}^s\}_{i=1}^{n_s}$ is the collection of $n_s$ images retrieved by Bing using the category name of the $k$-th class as keyword. As discussed in the next section, different methods will make different assumptions on the labels of the source examples.

We adopt a linear SVM as the model for the binary one-vs-the-rest classifiers. This choice is primarily motivated by the availability of several simple yet effective domain adaptation variants of SVM [5, 26], in addition to the aforementioned reasons of good performance and efficiency.

# 4 Methods

We now present the specific domain adaptation SVM algorithms. For brevity, we drop the subscript $k$ indicating dependence on the specific class. The hyperparameters $C$ of all classifiers are selected so as to minimize the multiclass cross validation error on the target training data. For all algorithms, we cope with the largely unequal number of positive and negative examples by normalizing the cost entries in the loss function by the respective class sizes.

## 4.1 Baselines: SVM$^s$, SVM$^t$, SVM$^{s \cup t}$

We include in our evaluation three algorithms *not* based on domain adaptation and use them as comparative baselines. We indicate with SVM$^t$ a linear SVM learned exclusively from the target examples. SVM$^s$ denotes an SVM learned from the source examples using the one-versus-the-rest scheme and assuming no outliers are present in the image search results. SVM$^{s \cup t}$ is a linear SVM trained on the union of the target and source examples. Specifically, for each class $k$, we train a binary SVM on the data obtained by merging $\mathcal{D}_k^t$ with $\mathcal{D}_k^s$, where the data in the latter set is assumed to contain only positive examples, i.e., no outliers. The hyperparameter $C$ is kept the same for all $K$ binary classifiers but tuned distinctly for each of the three methods by selecting the hyperparameter value yielding the best multiclass performance on the target training set (we used hold out validation on $\mathcal{D}_k^t$ for SVM$^s$ and 5-fold cross validation for both SVM$^t$ as well SVM$^{s \cup t}$).

## 4.2 Mixture of source and target hypotheses: MIXSVM

One of the simplest possible strategies for domain adaptation consists of using as final classifier a convex combination of the two SVM hypotheses learned independently from the source and target data. Despite its simplicity, this classifier has been shown to yield good empirical results [26].

Let us represent the source and target multiclass hypotheses as vector-valued functions $\boldsymbol{h}^s(\boldsymbol{f}) \to \mathbb{R}^K$, $\boldsymbol{h}^t(\boldsymbol{f}) \to \mathbb{R}^K$, where the $k$-th outputs are the respective SVM scores for class $k$. MIXSVM computes a convex combination $\boldsymbol{h}(\boldsymbol{f}) = \beta \boldsymbol{h}^s(\boldsymbol{f}) + (1-\beta)\boldsymbol{h}^t(\boldsymbol{f})$ and predicts the class $k^*$ associated to the largest output, i.e. $k^* = \arg\max_{k \in \{1,\ldots,K\}} h_k(\boldsymbol{f})$. The parameter $\beta \in [0,1]$ is determined via grid search by optimizing multiclass error on the target training set. We avoid biased estimates resulting from learning the hypothesis $\boldsymbol{h}^t$ and $\beta$ on the same training set by applying a two-stage procedure: we learn 5 distinct hypotheses $\boldsymbol{h}^t$ using 5-fold cross validation (with the hyperpameter value found for $\text{SVM}^t$) and compute prediction $\boldsymbol{h}^t(\boldsymbol{f}_i^t)$ at each training sample $\boldsymbol{f}_i^t$ using the cross validation hypothesis that was not trained on that example; we then use these predicted outputs to determine the optimal $\beta$. Last, we learn the final hypothesis $\boldsymbol{h}^t$ using the entire target training set.

### 4.3 Domain weighting: DWSVM

Another straightforward yet popular domain adaptation approach is to train a classifier using both the source and the target examples by weighting differently the two domains in the learning objective [5, 12, 4]. We follow the implementation proposed in [26] and weight the loss function values differently for the source and target examples by using two distinct SVM hyperparameters, $C_s$ and $C_t$, encoding the relative importance of the two domains. The values of these hyperparameters are selected by minimizing the multiclass 5-fold cross validation error on the target training set.

### 4.4 Feature augmentation: AUGSVM

We denote with AUGSVM the domain adaptation method described in [5]. The key-idea of this approach is to create a feature-augmented version of each individual example $\boldsymbol{f}$, where distinct feature augmentation mappings $\phi^s, \phi^t$ are used for the source and target data, respectively:

$$\phi^s(\boldsymbol{f}) = \begin{bmatrix} \boldsymbol{f}^T & \boldsymbol{f}^T & \boldsymbol{0}^T \end{bmatrix}^T \qquad \text{and} \qquad \phi^t(\boldsymbol{f}) = \begin{bmatrix} \boldsymbol{f}^T & \boldsymbol{0}^T & \boldsymbol{f}^T \end{bmatrix}^T, \tag{1}$$

where $\boldsymbol{0}$ indicates a $F$-dimensional vector of zeros. A linear SVM is then trained on the union of the feature-augmented source and target examples (using a single hyperparameter). The principle behind this mapping is that the SVM trained in the feature-augmented space has the ability to distinguish features having common behavior in the two domains (associated to the first $F$ SVM weights) from features having different properties in the two domains.

### 4.5 Transductive learning: TSVM

The previous methods implement different strategies to adjust the relative importance of the source and the training examples in the learning process. However, all these techniques assume that the source data is fully and correctly labeled. Unfortunately, in our practical problem this assumption is violated due to outliers and irrelevant results being present in the images retrieved by keyword search. To tackle this problem we propose to perform transductive inference on the label of the source data *during the learning*: the key-idea is to exploit the availability of strongly-labeled target training data to simultaneously determine the correct labels of the source training examples and incorporate this labeling information to improve the classifier. To address this task we employ the transductive SVM model introduced in [17]. Although this method is traditionally used to infer the labels of unlabeled data available at learning time, it outputs a proper inductive hypothesis and therefore can be used also to predict labels of unseen test examples. The problem of learning a transductive SVM in our context can be formulated as follows:

$$\min_{\boldsymbol{w}, \boldsymbol{y}^s} \frac{1}{2}||\boldsymbol{w}||^2 + C^t \sum_{i=1}^{N^t} c_i^t\, l(y_i^t \boldsymbol{w}^T \boldsymbol{f}_i^t) + \frac{C^s}{n^s} \sum_{j=1}^{n^s} l(y_j^s \boldsymbol{w}^T \boldsymbol{f}_j^s)$$

$$\text{subject to} \quad \frac{1}{n^s} \sum_{j=1}^{n^s} \max[0, \text{sign}(\boldsymbol{w}^T \boldsymbol{f}_j^s)] = \rho \tag{2}$$

where $l()$ denotes the loss function, $\boldsymbol{w}$ is the vector of SVM weights, $\boldsymbol{y}^s$ contains the labels of the source examples, and the $c_i^t$ are scalar coefficients used to counterbalance the effect of the unequal number of positive and negative examples: we set $c_i^t = 1/n^t$ if $y_i^t = 1$, $c_i^t = 1/((K-1)n^t)$ otherwise. The scalar parameter $\rho$ defines the fraction of source examples that we expect to be positive and is tuned via cross validation. Note that TSVM solves jointly for the separating hyperplane and the labels of the source examples by trading off maximization of the margin and minimization of the

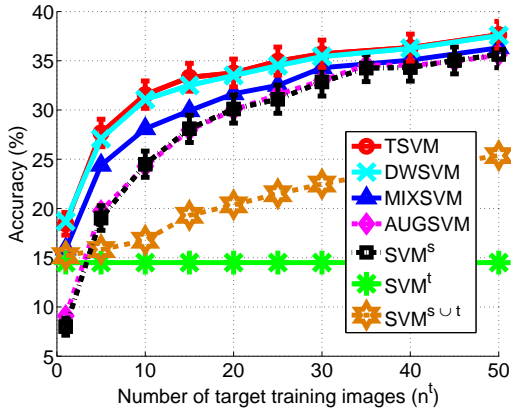

Figure 2: Recognition accuracy obtained with $n^s = 300$ Web photos and a varying number of Caltech256 target training examples.

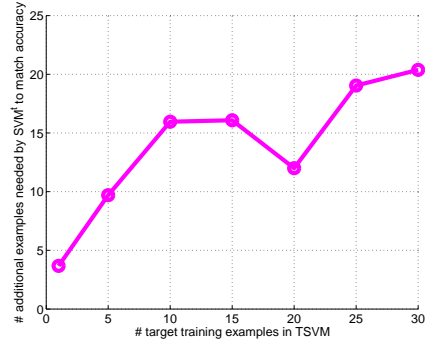

Figure 3: Manual annotation saving: the plot shows for a varying number of labeled examples given to TSVM the number of *additional* labeled images that would be needed by SVM$^t$ to achieve the same accuracy.

prediction errors on both source and target data. This optimization can be interpreted as implementing the cluster assumption, i.e., the expectation that points in a data cluster have the same label. We solve the optimization problem in Eq. 2 for a quadratic soft-margin loss function $l$ (i.e., $l$ is chosen to be the square of the hinge loss) using the minimization algorithm proposed in [27], which computes an efficient primal solution using the modified finite Newton method of [18]. This minimization approach is ideally suited to large-scale sparse data sets such as ours (about 70% of our features are zero). We used the same values of hyperparameters ($C^t$, $C^s$, and $\rho$) for all classes $k = 1, \ldots, K$ and selected them by minimizing the multiclass cross validation error. We also tried letting $\rho$ vary for each individual class but that led to slightly inferior results, possibly due to overfitting.

# 5 Experimental results

We now present the experimental results. Figure 2 shows the accuracy achieved by the different algorithms when using $n^s = 300$ and a varying number of training target examples ($n^t$). The accuracy is measured as the average of the mean recognition rate per class, using $m^t = 25$ test examples for each class. The best accuracy is achieved by the domain adaptation methods TSVM and DWSVM, which produce significant improvements over the SVM trained using only target examples (SVM$^t$), particularly for small values of $n^t$. For $n^t = 5$, TSVM yields a $65\%$ improvement over the best published results on this benchmark (for the same number of examples, an accuracy of $16.7\%$ is reported in [13]). Our method achieves this performance by analyzing additional images, the Internet photos, but since these are collected automatically and do not require any human supervision, the gain we achieve is effectively "human-cost free". It is interesting to note that while using solely source training images yields very low accuracy ($14.5\%$ for SVM$^s$), adding even just a single labeled target image produces a significant improvement (TSVM achieves $18.5\%$ accuracy with $n^t = 1$, and $27.1\%$ with $n^t = 5$): this indicates that the method can indeed adapt the classifier to work effectively on the target domain given a small amount of strongly-labeled data. It is interesting to note that while TSVM implements a form of outlier rejection as it solves for the labels of the source examples, DWSVM assumes that all source images in $\mathcal{D}_k^s$ are positive examples for class $k$. Yet, DWSVM achieves results similar to those of TSVM: this suggests that domain adaptation rather than outlier rejection is the key-factor contributing to the improvement with respect to the baselines.

By analyzing the performance of the baselines in figure 2 we observe that training exclusively with Web images (SVM$^s$) yields much lower accuracy than using strongly-labeled data (SVM$^t$): this is consistent with prior work [9, 29]. Furthermore, the poor accuracy of SVM$^{s \cup t}$ compared to SVM$^t$ suggests that naïvely adding a large number of source examples to the target training set without consideration of the domain differences not only does not help but actually worsens the recognition.

Figure 3 illustrates the significant manual annotation saving produced by our approach: the $x$-axis is the number of target labeled images provided to TSVM while the $y$-axis shows the number of *additional* labeled examples that would be needed by SVM$^t$ to achieve the same accuracy.

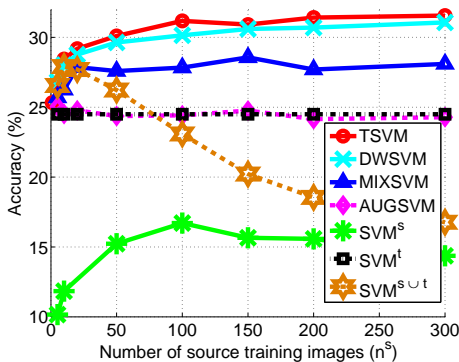
Figure 4: Classification accuracy of the different methods using $n^t = 10$ target training images and a varying number of source examples.

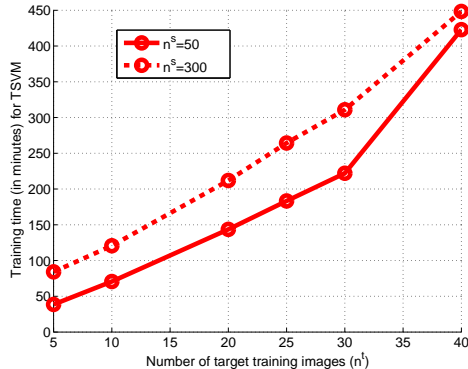
Figure 5: Training time: time needed to learn a multiclass classifier for Caltech256 using TSVM.

The setting $n^s = 300$ in the results above was chosen by studying the recognition accuracy as a function of the number of source examples: we carried out an experiment where we fixed the number $n^t$ of target training example for each category to an intermediate value ($n^t = 10$), and varied the number $n^s$ of top image results used as source training examples for each class. Figure 4 summarizes the results. We notice that the performance of the SVM trained only on source images (SVM$^s$) peaks at $n^s = 100$ and decreases monotonically after this value. This result can be explained by observing that image search engines provide images sorted according to estimated relevancy with respect to the keyword. It is conceivable to assume that images far down in the ranking list will often tend to be outliers, which may lead to degradation of recognition particularly for non-robust models. Despite this, we see that the domain adaptation methods TSVM and DWSVM exhibit a monotonically non-decreasing accuracy as $n^s$ grows: this indicates that these methods are highly robust to outliers and can make effective use of source data even when increasing $n^s$ causes a likely decrease of the fraction of inliers and relevant results. Contrast these robust performances with the accuracy of SVM$^{s \cup t}$, which grows as we begin adding source examples but then decays rapidly after $n^s = 10$ and approaches the poor recognition of SVM$^s$ for large values of $n^s$.

Our approach compares very favorably with competing algorithms also in terms of computational complexity: training TSVM (without cross validation) on Caltech256 with $n^t = 5$ and $n^s = 300$ takes 84 minutes on a AMD Opteron Processor 280 2.4GHz; training the multiclass method of [13] using 5 labeled examples per class takes about 23 hours on the same machine (for fairness of comparison, we excluded cross validation even for this method). A detailed analysis of training time as a function of the number of labeled training examples is reported in figure 5. Evaluation of our model on a test example takes 0.18ms, while the method of [13] requires 37ms.

## 6   Discussion and future work

In this work we have investigated the application of domain adaptation methods to object categorization using Web photos as source data. Our analysis indicates that, while object classifiers learned exclusively from Web data are inferior to fully-supervised models, the use of domain adaptation methods to combine Web photos with small amounts of strongly labeled data leads to state-of-the-art results. The proposed strategy should be particularly useful in scenarios where labeled data is scarce or expensive to acquire. Future work will include application of our approach to combine data from multiple source domains (e.g., images obtained from different search engines or photo sharing sites) and different media (e.g., text and video). Additional material including software and our source training data may be obtained from [1].

### Acknowledgments

We are grateful to Andrew Fitzgibbon and Martin Szummer for discussion. We thank Vikas Sindhwani for providing code. This research was funded in part by NSF CAREER award IIS-0952943.

## Footnotes

[1]Note that image search results may have changed since these examples were captured.

# References

[1] `http://vlg.cs.dartmouth.edu/projects/domainadapt`.

[2] T. L. Berg and D. A. Forsyth. Animals on the web. In *CVPR*, pages 1463–1470, 2006.

[3] I. Bierderman. Recognition-by-components: A theory of human image understanding. *Psychological Review*, 94(2):115–147, 1987.

[4] J. Blitzer, K. Crammer, A. Kulesza, F. Pereira, and J. Wortman. Learning bounds for domain adaptation. In *NIPS*, 2007.

[5] H. Daume III. Frustratingly easy domain adaptation. In *ACL*, 2007.

[6] J. Deng, W. Dong, R. Socher, L.-J. Li, K. Li, and L. Fei-Fei. ImageNet: A Large-Scale Hierarchical Image Database. In *CVPR*, 2009.

[7] M. Everingham, L. Van Gool, C. K. I. Williams, J. Winn, and A. Zisserman. The PASCAL Visual Object Classes Challenge 2010 (VOC2010) Results.

[8] L. Fei-Fei, R. Fergus, and P. Perona. One-shot learning of object categories. *IEEE Trans. Pattern Anal. Mach. Intell.*, 28(4):594–611, 2006.

[9] R. Fergus, L. Fei-Fei, P. Perona, and A. Zisserman. Learning object categories from google's image search. In *ICCV*, pages 1816–1823, 2005.

[10] R. Fergus, P. Perona, and A. Zisserman. A visual category filter for google images. In *ECCV*, 2004.

[11] R. Fergus, Y. Weiss, and A. Torralba. Semi-supervised learning in gigantic image collections. In Y. Bengio, D. Schuurmans, J. Lafferty, C. K. I. Williams, and A. Culotta, editors, *NIPS 22*, 2009.

[12] J. R. Finkel and C. D. Manning. Hierarchical bayesian domain adaptation. In *Proceedings of the North American Association of Computational Linguistics (NAACL 2009)*, 2009.

[13] P. V. Gehler and S. Nowozin. On feature combination for multiclass object classification. In *IEEE International Conference on Computer Vision (ICCV)*, 2009.

[14] G. Griffin, A. Holub, and P. Perona. Caltech-256 object category dataset. Technical Report 7694, California Institute of Technology, 2007.

[15] A. Holub, M. Welling, and P. Perona. Exploiting unlabelled data for hybrid object classification. In *NIPS, Interclass transfer workshop*, 2005.

[16] J. Huang, A. J. Smola, A. Gretton, K. M. Borgwardt, and B. Schölkopf. Correcting sample selection bias by unlabeled data. In *NIPS*, pages 601–608, 2006.

[17] T. Joachims. Transductive inference for text classification using support vector machines. In *ICML*, pages 200–209, 1999.

[18] S. S. Keerthi and D. DeCoste. A modified finite newton method for fast solution of large scale linear svms. *Journal of Machine Learning Research*, 6:341–361, 2005.

[19] C. Leistner, H. Grabner, and H. Bischof. Semi-supervised boosting using visual similarity learning. In *CVPR*, 2008.

[20] L. Li and L. Fei-Fei. Optimol: Automatic online picture collection via incremental model learning. *Intl. Jrnl. of Computer Vision*, 88(2):147–168, 2010.

[21] E. G. Miller, N. E. Matsakis, and P. A. Viola. Learning from one example through shared densities on transforms. In *CVPR*, 2000.

[22] A. Quattoni, M. Collins, and T. Darrell. Transfer learning for image classification with sparse prototype representations. In *CVPR*, 2008.

[23] B. C. Russell, A. B. Torralba, K. P. Murphy, and W. T. Freeman. Labelme: A database and web-based tool for image annotation. *International Journal of Computer Vision*, 77(1-3):157–173, 2008.

[24] K. Saenko, B. Kulis, M. Fritz, and T. Darrell. Adapting visual category models to new domains. In *European Conference on Computer Vision (ECCV)*, Sept. 2010.

[25] F. Schroff, A. Criminisi, and A. Zisserman. Harvesting image databases from the web. In *ICCV*, 2007.

[26] G. Schweikert, C. Widmer, B. Schölkopf, and G. Rätsch. An empirical analysis of domain adaptation algorithms for genomic sequence analysis. In *NIPS*, pages 1433–1440, 2008.

[27] V. Sindhwani and S. S. Keerthi. Large scale semi-supervised linear svms. In *SIGIR*, pages 477–484, 2006.

[28] L. Torresani, M. Szummer, and A. Fitzgibbon. Efficient object category recognition using classemes. In *European Conference on Computer Vision (ECCV)*, pages 776–789, Sept. 2010.

[29] S. Vijayanarasimhan and K. Grauman. Keywords to visual categories: Multiple-instance learning for weakly supervised object categorization. In *CVPR*, 2008.

[30] L. von Ahn. Games with a purpose. *IEEE Computer*, 39(6):92–94, 2006.

